# Budgeted Optimization with Concurrent Stochastic-Duration Experiments

**Javad Azimi, Alan Fern, Xiaoli Z. Fern**
School of EECS, Oregon State University
{azimi, afern, xfern}@eecs.oregonstate.edu

## Abstract

Budgeted optimization involves optimizing an unknown function that is costly to evaluate by requesting a limited number of function evaluations at intelligently selected inputs. Typical problem formulations assume that experiments are selected one at a time with a limited total number of experiments, which fail to capture important aspects of many real-world problems. This paper defines a novel problem formulation with the following important extensions: 1) allowing for concurrent experiments; 2) allowing for stochastic experiment durations; and 3) placing constraints on both the total number of experiments and the total experimental time. We develop both offline and online algorithms for selecting concurrent experiments in this new setting and provide experimental results on a number of optimization benchmarks. The results show that our algorithms produce highly effective schedules compared to natural baselines.

## 1   Introduction

We study the optimization of an unknown function $f$ by requesting $n$ experiments, each specifying an input $x$ and producing a noisy observation of $f(x)$. In practice, the function $f$ might be the performance of a device parameterized by $x$. We consider the setting where running experiments is costly (e.g. in terms of time), which renders methods that rely on many function evaluations, such as stochastic search or empirical gradient methods, impractical. Bayesian optimization (BO) [8, 4] addresses this issue by leveraging Bayesian modeling to maintain a posterior over the unknown function based on previous experiments. The posterior is then used to intelligently select new experiments to trade-off exploring new parts of the experimental space and exploiting promising parts.

Traditional BO follows a sequential approach where only one experiment is selected and run at a time. However, it is often desirable to select more than one experiment at a time so that multiple experiments can be run simultaneously to leverage parallel facilities. Recently, Azimi et al. (2010) proposed a batch BO algorithm that selects a batch of $k \geq 1$ experiments at a time. While this broadens the applicability of BO, it is still limited to selecting a fixed number of experiments at each step. As such, prior work on BO, both batch and sequential, completely ignores the problem of how to schedule experiments under fixed experimental budget and time constraints. Furthermore, existing work assumes that the durations of experiments are identical and deterministic, whereas in practice they are often stochastic.

Consider one of our motivating applications of optimizing the power output of nano-enhanced Microbial Fuel Cells (MFCs). MFCs [3] use micro-organisms to generate electricity. Their performance depends

strongly on the surface properties of the anode [10]. Our problem involves optimizing nano-enhanced anodes, where various types of nano-structures, e.g. carbon nano-wire, are grown directly on the anode surface. Because there is little understanding of how different nano-enhancements impact power output, optimizing anode design is largely guess work. Our original goal was to develop BO algorithms for aiding this process. However, many aspects of this domain complicate the application of BO. First, there is a fixed budget on the number of experiments that can be run due to limited funds and a fixed time period for the project. Second, we can run multiple concurrent experiments, limited by the number of experimental apparatus. Third, the time required to run each experiment is variable because each experiment requires the construction of a nano-structure with specific properties. Nano-fabrication is highly unpredictable and the amount of time to successfully produce a structure is quite variable. Clearly prior BO models fail to capture critical aspects of the experimental process in this domain.

In this paper, we consider the following extensions. First, we have $l$ available labs (which may correspond to experimental stations at one location or to physically distinct laboratories), allowing up to $l$ concurrent experiments. Second, experiments have stochastic durations, independently and identically distributed according to a known density function $p_d$. Finally, we are constrained by a budget of $n$ total experiments and a time horizon $h$ by which point we must finish. The goal is to maximize the unknown function $f$ by selecting experiments and when to start them while satisfying the constraints.

We propose offline (Section 4) and online (Section 5) scheduling approaches for this problem, which aim to balance two competing factors. First, a scheduler should ensure that all $n$ experiments complete within the horizon $h$, which encourages high concurrency. Second, we wish to select new experiments given as many previously completed experiments as possible to make more intelligent experiment selections, which encourages low concurrency. We introduce a novel measure of the second factor, cumulative prior experiments (CPE) (Section 3), which our approaches aim to optimize. Our experimental results indicate that these approaches significantly outperform a set of baselines across a range of benchmark optimization problems.

## 2 Problem Setup

Let $\mathcal{X} \subseteq \Re^d$ be a $d$-dimensional compact input space, where each dimension $i$ is bounded in $[a_i, b_i]$. An element of $\mathcal{X}$ is called an *experiment*. An unknown real-valued function $f : \mathcal{X} \to \Re$ represents the expected value of the dependent variable after running an experiment. For example, $f(x)$ might be the result of a wet-lab experiment described by $x$. Conducting an experiment $x$ produces a noisy outcome $y = f(x) + \epsilon$, where $\epsilon$ is a random noise term. Bayesian Optimization (BO) aims to find an experiment $x \in \mathcal{X}$ that approximately maximizes $f$ by requesting a limited number of experiments and observing their outcomes.

We extend traditional BO algorithms and study the experiment scheduling problem. Assuming a known density function $p_d$ for the experiment durations, the inputs to our problem include the total number of available labs $l$, the total number of experiments $n$, and the time horizon $h$ by which we must finish. The goal is to design a policy $\pi$ for selecting when to start experiments and which ones to start to optimize $f$. Specifically, the inputs to $\pi$ are the set of completed experiments and their outcomes, the set of currently running experiments with their elapsed running time, the number of free labs, and the remaining time till the horizon. Given this information, $\pi$ must select a set of experiments (possibly empty) to start that is no larger than the number of free labs. Any run of the policy ends when either $n$ experiments are completed or the time horizon is reached, resulting in a set $X$ of $n$ or fewer completed experiments. The objective is to obtain a policy with small *regret*, which is the expected difference between the optimal value of $f$ and the value of $f$ for the predicted best experiment in $X$. In theory, the optimal policy can be found by solving a POMDP with hidden state corresponding to the unknown function $f$. However, this POMDP is beyond the reach of any existing solvers. Thus, we focus on defining and comparing several principled policies that work well in practice, but without optimality guarantees. Note that this problem has not been studied in the literature to the best of our knowledge.

## 3 Overview of General Approach

A policy for our problem must make two types of decisions: 1) scheduling when to start new experiments, and 2) selecting the specific experiments to start. In this work, we factor the problem based on these decisions and focus on approaches for scheduling experiments. We assume a black box function *SelectBatch* for intelligently selecting the $k \geq 1$ experiments based on both completed and currently running experiments. The implementation of *SelectBatch* is described in Section 6.

Optimal scheduling to minimize regret appears to be computationally hard for non-trivial instances of *SelectBatch*. Further, we desire scheduling approaches that do not depend on the details of *SelectBatch*, but work well for any reasonable implementation. Thus, rather than directly optimizing regret for a specific *SelectBatch*, we consider the following surrogate criteria. First, we want to finish all $n$ experiments within the horizon $h$ with high probability. Second, we would like to select each experiment based on as much information as possible, measured by the number of previously completed experiments. These two goals are at odds, since maximizing the completion probability requires maximizing concurrency of the experiments, which minimizes the second criterion. Our offline and online scheduling approaches provide different ways for managing this trade-off.

To quantify the second criterion, consider a complete execution $E$ of a scheduler. For any experiment $e$ in $E$, let $prior_E(e)$ denote the number of experiments in $E$ that *completed* before starting $e$. We define the *cumulative prior experiments (CPE)* of $E$ as: $\sum_{e \in E} prior_E(e)$. Intuitively, a scheduler with a high expected CPE is desirable, since CPE measures the total amount of information *SelectBatch* uses to make its decisions.

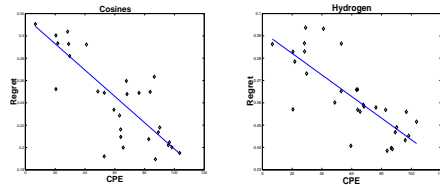

Figure 1: The correlation between CPE and regret for 30 different schedulers on two BO benchmarks.

CPE agrees with intuition when considering extreme policies. A poor scheduler that starts all $n$ experiments at the same time (assuming enough labs) will have a minimum CPE of zero. Further, CPE is maximized by a scheduler that sequentially executes all experiments (assuming enough time). However, in between these extremes, CPE fails to capture certain intuitive properties. For example, CPE increases linearly in the number of prior experiments, while one might expect diminishing returns as the number of prior experiments becomes large. Similarly, as the number of experiments started together (the batch size) increases, we might also expect diminishing returns since *SelectBatch* must choose the experiments based on the same prior experiments. Unfortunately, quantifying these intuitions in a general way is still an open problem. Despite its potential shortcomings, we have found CPE to be a robust measure in practice.

To empirically examine the utility of CPE, we conducted experiments on a number of BO benchmarks. For each domain, we used 30 manually designed diverse schedulers, some started more experiments early on than later, and vice-versa, while others included random and uniform schedules. We measured the average regret achieved for each scheduler given the same inputs and the expected CPE of the executions. Figure 1 shows the results for two of the domains (other results are highly similar), where each point corresponds to the average regret and CPE of a particular scheduler. We observe a clear and non-trivial correlation between regret and CPE, which provides empirical evidence that CPE is a useful measure to optimize. Further, as we will see in our experiments, the performance of our methods is also highly correlated with CPE.

## 4 Offline Scheduling

We now consider offline schedules, which assign *start times* to all $n$ experiments before the experimental process begins. Note that while the schedules are offline, the overall BO policy has online characteristics, since the exact experiments to run are only specified when they need to be started by *SelectBatch*, based

on the most recent information. This offline scheduling approach is often convenient in real experimental domains where it is useful to plan out a static equipment/personnel schedule for the duration of a project. Below we first consider a restricted class of schedules, called staged schedules, for which we present a solution that optimizes CPE. Next, we describe an approach for a more general class of schedules.

## 4.1 Staged Schedules

A *staged schedule* defines a consecutive sequence of $N$ experimental stages, denoted by a sequence of tuples $\langle (n_i, d_i) \rangle_{i=1}^{N}$, where $0 < n_i \leq l$, $\sum_i d_i \leq h$, and $\sum_i n_i \leq n$. Stage $i$ begins by starting up $n_i$ new experiments selected by *SelectBatch* using the most recent information, and ends after a duration of $d_i$, upon which stage $i+1$ starts. In some applications, staged schedules are preferable as they allow project planning to focus on a relatively small number of time points (the beginning of each stage). While our approach tries to ensure that experiments finish within their stage, experiments are never terminated and hence might run longer than their specified duration. If, because of this, at the beginning of stage $i$ there are not $n_i$ free labs, the experiments will wait till labs free up.

We say that an execution $E$ of a staged schedule $S$ is *safe* if each experiment is completed within its specified duration in $S$. We say that a staged schedule $S$ is *p-safe* if with probability at least $p$ an execution of $S$ is safe which provides a probabilistic guarantee that all $n$ experiments complete within the horizon $h$. Further, it ensures with probability $p$ that the maximum number of concurrent experiments when executing $S$ is $\max_i n_i$ (since experiments from two stages will not overlap with probability $p$). As such, we are interested in finding staged schedules that are $p$-safe for a user specified $p$, e.g. 95%. Meanwhile, we want to maximize CPE. The CPE of any safe execution of $S$ (slightly abusing notation) is: $\text{CPE}(S) = \sum_{i=2}^{N} n_i \sum_{j=1}^{i-1} n_j$. Typical applications will use relative high values of $p$, since otherwise experimental resources would be wasted, and thus with high probability we expect the CPE of an execution of $S$ to equal $\text{CPE}(S)$.

Our goal is thus to maximize $\text{CPE}(S)$ while ensuring $p$-safeness. It turns out that for any fixed number of stages $N$, the schedules that maximize $\text{CPE}(S)$ must be *uniform*. A staged schedule is defined to be *uniform* if $\forall i, j, |n_i - n_j| \leq 1$, i.e., the batch sizes across stages may differ by at most a single experiment.

**Proposition 1.** *For any number of experiments $n$ and labs $l$, let $\mathcal{S}_N$ be the set of corresponding $N$ stage schedules, where $N \geq \lceil n/l \rceil$. For any $S \in \mathcal{S}_N$, $CPE(S) = \max_{S' \in \mathcal{S}_N} CPE(S')$ if and only if $S$ is uniform.*

It is easy to verify that for a given $n$ and $l$, an $N$ stage uniform schedule achieves a strictly higher CPE than any $N - 1$ stage schedule. This implies that we should prefer uniform schedules with maximum number of stages allowed by the $p$-safeness restriction. This motivates us to solve the following problem: *Find a p-safe uniform schedule with maximum number of stages.*

Our approach, outlined in Algorithm 1, considers $N$ stage schedules in order of increasing $N$, starting at the minimum possible number of stages $N = \lceil n/l \rceil$ for running all experiments. For each value of $N$, the call to *MaxProbUniform* computes a uniform schedule $S$ with the highest probability of a safe execution, among all $N$ stage uniform schedules. If the resulting schedule is $p$-safe then we consider $N + 1$ stages. Otherwise, there is no uniform $N$ stage schedule that is $p$-safe and

---

**Algorithm 1** Algorithm for computing a $p$-safe uniform schedule with maximum number of stages.

---

**Input:** number of experiments $(n)$, number of labs $(l)$, horizon $(h)$, safety probability $(p)$
**Output:** A $p$-safe uniform schedule with maximum number of stages

    $N = \lceil n/l \rceil$, $S \leftarrow$ null
    **loop**
        $S' \leftarrow \text{MaxProbUniform}(N, n, l, h)$
        **if** $S'$ is not $p$-safe **then**
            **return** $S$
        **end if**
        $S \leftarrow S'$, $N \leftarrow N + 1$
    **end loop**

---

we return a uniform $N - 1$ stage schedule, which was computed in the previous iteration.

It remains to describe the *MaxProbUniform* function, which computes a uniform $N$ stage schedule $S = \langle (n_i, d_i) \rangle_{i=1}^N$ that maximizes the probability of a safe execution. First, any $N$ stage uniform schedule must have $N' = (n \mod N)$ stages with $n' = \lfloor n/N \rfloor + 1$ experiments and $N - N'$ stages with $n' - 1$ experiments. Furthermore, the probability of a safe execution is invariant to the ordering of the stages, since we assume i.i.d. distribution on the experiment durations. The *MaxProbUniform* problem is now reduced to computing the durations $d_i$ of $S$ that maximize the probability of safeness for each given $n_i$. For this we will assume that the distribution of the experiment duration $p_d$ is log-concave, which allows us to characterize the solution using the following lemma.

**Lemma 1.** *For any duration distribution $p_d$ that is log-concave, if an $N$ stage schedule $S = \langle (n_i, d_i) \rangle_{i=1}^N$ is p-safe, then there is a p-safe $N$ stage schedule $S' = \langle (n_i, d_i') \rangle_{i=1}^N$ such that if $n_i = n_j$ then $d_i' = d_j'$.*

This lemma suggests that any stages with equal $n_i$'s should have equal $d_i$'s to maximize the probability of safe execution. For a uniform schedule, $n_i$ is either $n'$ or $n' - 1$. Thus we only need to consider schedules with two durations, $d'$ for stages with $n_i = n'$ and $d''$ for stages with $n_i = n' - 1$. Since all durations must sum to $h$, $d'$ and $d''$ are deterministically related by: $d'' = \frac{h - d' \cdot N'}{N - N'}$. Based on this, for any value of $d'$ the probability of the uniform schedule using durations $d'$ and $d''$ is as follows, where $P_d$ is the CDF of $p_d$.

$$[P_d(d')]^{N' \cdot n'} \left[ P_d \left( \frac{h - d' \cdot N'}{N - N'} \right) \right]^{(N-N') \cdot (n'-1)} \tag{1}$$

We compute *MaxProbUniform* by maximizing Equation 1 with respect to $d'$ and using the corresponding duration for $d''$. Putting everything together we get the following result.

**Theorem 1.** *For any log-concave $p_d$, computing* MaxProbUniform *by maximizing Equation 1 over $d'$, if a p-safe uniform schedule exists, Algorithm 1 returns a maximum-stage p-safe uniform schedule.*

### 4.2 Independent Lab Schedules

We now consider a more general class of offline schedules and a heuristic algorithm for computing them. This class allows the start times of different labs to be decoupled, desirable in settings where labs are run by independent experimenters. Further, our online scheduling approach is based on repeatedly calling an offline scheduler, which requires the flexibility to make schedules for labs in different stages of execution.

An *independent lab (IL) schedule* $S$ specifies a number of labs $k < l$ and for each lab $i$, a number of experiments $m_i$ such that $\sum_i m_i = n$. Further, for each lab $i$ a sequence of $m_i$ durations $D_i = \langle d_i^1, \ldots, d_i^{m_i} \rangle$ is given. The execution of $S$ runs each lab independently, by having each lab start up experiments whenever they move to the next stage. Stage $j$ of lab $i$ ends after a duration of $d_i^j$, or after the experiment finishes when it runs longer than $d_i^j$ (i.e. we do not terminate experiments). Each experiment is selected according to SelectBatch, given information about all completed and running experiments across all labs.

We say that an execution of an IL schedule is safe if all experiments finish within their specified durations, which also yields a notion of $p$-safeness. We are again interested in computing $p$-safe schedules that maximizes the CPE. Intuitively, CPE will be maximized if the amount of concurrency during an execution is minimized, suggesting the use of as few labs as possible. This motivates the problem of finding a $p$-safe IL schedule that use the minimum number of labs. Below we describe our heuristic approach to this problem.

**Algorithm Description.** Starting with $k = 1$, we compute a $k$ labs IL schedule with the goal of maximizing the probability of safe execution. If this probability is less than $p$, we increment $k$, and otherwise output the schedule for $k$ labs. To compute a schedule for each value of $k$, we first allocate the number of experiments $m_i$ across $k$ labs as uniformly as possible. In particular, $(n \mod k)$ labs will have $\lfloor n/k \rfloor + 1$ experiments and $k - (n \mod k)$ labs will have $\lfloor n/k \rfloor$ experiments. This choice is motivated by the intuition that the best way to maximize the probability of a safe execution is to distribute the work across labs as uniformly as possible. Given $m_i$ for each lab, we assign all durations of lab $i$ to be $h/m_i$, which can be shown to be optimal for log-concave $p_d$. In this way, for each value of $k$ the schedule we compute has just two possible values of $m_i$ and labs with the same $m_i$ have the same stage durations.

## 5 Online Scheduling Approaches

We now consider online scheduling, which selects the start time of experiments online. The flexibility of the online approaches offers the potential to outperform offline schedules by adapting to specific stochastic outcomes observed during experimental runs. Below we first describe two baseline online approaches, followed by our main approach, policy switching, which aims to directly optimize CPE.

**Online Fastest Completion Policy (OnFCP).** This baseline policy simply tries to finish all of the $n$ experiments as quickly as possible. As such, it keeps all $l$ labs busy as long as there are experiments left to run. Specifically whenever a lab (or labs) becomes free the policy immediately uses SelectBatch with the latest information to select new experiments to start right away. This policy will achieve a low value of expected CPE since it maximizes concurrency.

**Online Minimum Eager Lab Policy (OnMEL).** One problem with OnFCP is that it does not attempt to use the full time horizon. The OnMEL policy simply restricts OnFCP to use only $k$ labs, where $k$ is the minimum number of labs required to guarantee with probability at least $p$ that all $n$ experiments complete within the horizon. Monte-Carlo simulation is used to estimate $p$ for each $k$.

**Policy Switching (PS).** Our policy switching approach decides the number of new experiments to start at each decision epoch. Decision epochs are assumed to occur every $\Delta$ units of time, where $\Delta$ is a small constant relative to the expected experiment durations. The motivation behind policy switching is to exploit the availability of a policy generator that can produce multiple policies at any decision epoch, where at least one of them is expected to be good. Given such a generator, the goal is to define a new (switching) policy that performs as well or better than the best of the generated policies in any state. In our case, the objective is to improve CPE, though other objectives can also be used. This is motivated by prior work on policy switching [6] over a fixed policy library, and generalize that work to handle arbitrary policy generators instead of static policy libraries. Below we describe the general approach and then the specific policy generator that we use.

Let $t$ denote the number of remaining decision epochs (stages-to-go), which is originally equal to $\lfloor h/\Delta \rfloor$ and decremented by one each epoch. We use $s$ to denote the experimental state of the scheduling problem, which encodes the number of completed experiments and ongoing experiments with their elapsed running time. We assume access to a *policy generator* $\Pi(s, t)$ which returns a set of base scheduling policies (possibly non-stationary) given inputs $s$ and $t$. Prior work on policy switching [6] corresponds to the case where $\Pi(s, t)$ returns a fixed set of policies regardless of $s$ and $t$. Given $\Pi(s, t)$, $\bar{\pi}(s, t, \pi)$ denotes the resulting switching policy based on $s$, $t$, and the base policy $\pi$ selected in the previous epoch. The decision returned by $\bar{\pi}$ is computed by first conducting $N$ simulations of each policy returned by $\Pi(s, t)$ along with $\pi$ to estimate their CPEs. The base policy with the highest estimated CPE is then selected and its decision is returned by $\bar{\pi}$. The need to compare to the previous policy $\pi$ is due to the use of a dynamic policy generator, rather than a fixed library. The base policy passed into policy switching for the first decision epoch can be arbitrary.

Despite its simplicity, we can make guarantees about the quality of $\bar{\pi}$ assuming a bound on the CPE estimation error. In particular, the CPE of the switching policy will not be much worse than the best of the policies produced by our generator given accurate simulations. We say that a CPE estimator is $\epsilon$-accurate if it can estimate the CPE $C_t^\pi(s)$ of any base policy $\pi$ for any $s$ and $t$ within an accuracy bound of $\epsilon$. Below we denote the expected CPE of $\bar{\pi}$ for $s$, $t$, and $\pi$ to be $C_t^{\bar{\pi}}(s, \pi)$.

**Theorem 2.** *Let $\Pi(s, t)$ be a policy generator and $\bar{\pi}$ be the switching policy computed with $\epsilon$-accurate estimates. For any state $s$, stages-to-go $t$, and base policy $\pi$, $C_t^{\bar{\pi}}(s, \pi) \geq \max_{\pi' \in \Pi(s,t) \cup \{\pi\}} C_t^{\pi'}(s) - 2t\epsilon$.*

We use a simple policy generator $\Pi(s, t)$ that makes multiple calls to the offline IL scheduler described earlier. The intuition is to notice that the produced $p$-safe schedules are fairly pessimistic in terms of the experiment runtimes. In reality many experiments will finish early and we can adaptively exploit such situations. Specifically, rather than follow the fixed offline schedule we may choose to use fewer labs and hence improve CPE. Similarly if experiments run too long, we will increase the number of labs.

Table 1: Benchmark Functions

| Cosines(2)[1] | $1 - (u^2 + v^2 - 0.3cos(3\pi u) - 0.3cos(3\pi v))$ $u = 1.6x - 0.5, v = 1.6y - 0.5$ | Rosenbrock(2)[1] | $10 - 100(y - x^2)^2 - (1 - x)^2$ |
|---|---|---|---|
| Hartman(3,6)[7] | $\Sigma_{i=1} 4\alpha_i \exp\left[-\Sigma_{j=1}^d A_{ij}(x_j - P_{ij})^2\right]$ $\alpha_{1\times4},\ A_{4\times d},\ P_{4\times d}$ are constants | Michalewicz(5)[9] | $-\sum_{i=1}^5 \sin(x_i) \cdot \left(\sin\left(\frac{i.x_i^2}{\pi}\right)\right)^{20}$ |
| Shekel(4)[7] | $\Sigma_{i=1}^{10} \frac{1}{\alpha_i + \Sigma_{j=1} 4(x_j - A_{ji})^2}$ | $\alpha_{1\times10},\ A_{4\times10}$ are constants | |

We define $\Pi(s,t)$ to return $k + 1$ policies, $\{\pi_{(s,t,0)}, \ldots, \pi_{(s,t,k)}\}$, where $k$ is the number of experiments running in $s$. Policy $\pi_{(s,t,i)}$ is defined so that it waits for $i$ current experiments to finish, and then uses the offline IL scheduler to return a schedule. This amounts to adding a small lookahead to the offline IL scheduler where different amounts of waiting time are considered [1]. Note that the definition of these policies depends on $s$ and $t$ and hence can not be viewed as a fixed set of static policies as used by traditional policy switching. In the initial state $s_0$, $\pi_{(s_0,h,0)}$ corresponds to the offline IL schedule and hence the above theorem guarantees that we will not perform much worse than the offline IL, with the expectation of performing much better. Whenever policy switching selects a $\pi_i$ with $i > 0$ then no new experiments will be started and we wait for the next decision epoch. For $i = 0$, it will apply the offline IL scheduler to return a $p$-safe schedule to start immediately, which may require starting new labs to ensure high probability of completing $n$ experiments.

## 6  Experiments

**Implementation of *SelectBatch*.** Given the set of completed experiments $\mathcal{O}$ and on-going experiments $\mathcal{A}$, SelectBatch selects $k$ new experiments. We implement SelectBatch based on a recent batch BO algorithm [2], which greedily selects $k$ experiments considering only $\mathcal{O}$. We modify this greedy algorithm to also consider $\mathcal{A}$ by forcing the selected batch to include the ongoing experiments plus $k$ additional experiments. SelectBatch makes selections based on a posterior over the unknown function $f$. We use Gaussian Process with the RBF kernel and the kernel width $= 0.01 \sum_{i=1}^d l_i$, where $l_i$ is the input space length in dimension $i$.

**Benchmark Functions.** We evaluate our scheduling policies using 6 well-known synthetic benchmark functions (shown in Tab. 1 with dimension inside the parenthesis) and two real-world benchmark functions *Hydrogen* and *FuelCell* over $[0,1]^2$ [2]. The Hydrogen data is produced by a study on biosolar hydrogen production [5], where the goal was to maximize hydrogen production of a particular bacteria by optimizing PH and Nitrogen levels. The FuelCell data was collected in our motivating application mentioned in Sect. 1. In both cases, the benchmark function was created by fitting regression models to the available data.

**Evaluation.** We consider a $p$-safeness guarantee of $p = 0.95$ and the number of available labs $l$ is 10. For $p_d(x)$, we use one sided *truncated normal distribution* such that $x \in (0, \inf)$ with $\mu = 1$, $\sigma^2 = 0.1$, and we set the total number of experiments $n = 20$. We consider three time horizons $h$ of 6, 5, and 4.

Given $l$, $n$ and $h$, to evaluate policy $\pi$ using function $f$ (with a set of initial observed experiments), we execute $\pi$ and get a set $X$ of $n$ or fewer completed experiments. We measure the regret of $\pi$ as the difference between the optimal value of $f$ (known for all eight functions) and the $f$ value of the predicted best experiment in $X$.

**Results.** Table 2 shows the results of our proposed offline and online schedulers. We also include, as a reference point, the result of the *un-constrained* sequential policy (i.e., selecting one experiment at a time) using SelectBatch, which can be viewed as an effective *upper bound* on the optimal performance of any constrained scheduler because it ignores the time horizon ($h = \infty$). The values in the table correspond to the regrets (smaller values are better) achieved by each policy, averaged across 100 independent runs with the same initial experiments (5 for 2-d and 3-d functions and 20 for the rest) for all policies in each run.

Table 2: The proposed policies results for different horizons.

| Function | $h=\infty$ | OnFCP | OfStaged | OfIL | OnMEL | PS | OfStaged | OfIL | OnMEL | PS | OfStaged | OfIL | OnMEL | PS |
|---|---|---|---|---|---|---|---|---|---|---|---|---|---|---|
| | | | h=4 | | | | h=5 | | | | h=6 | | | |
| Cosines | .142 | .339 | .181 | .195 | .275 | .205 | .181 | .194 | .274 | .150 | .167 | .147 | .270 | .156 |
| FuelCell | .160 | .240 | .182 | .191 | .258 | .206 | .167 | .190 | .239 | .185 | .154 | .163 | .230 | .153 |
| Hydro | .025 | .115 | .069 | .070 | .123 | .059 | .071 | .069 | .086 | .042 | .036 | .035 | .064 | .025 |
| Rosen | .008 | .013 | .010 | .009 | .013 | .008 | .009 | .008 | .011 | .008 | .007 | .009 | .010 | .009 |
| Hart(3) | .037 | .095 | .070 | .069 | .096 | .067 | .055 | .064 | .081 | .045 | .045 | .050 | .070 | .038 |
| Michal | .465 | .545 | .509 | .508 | .525 | .502 | .500 | .510 | .521 | .494 | .477 | .460 | .502 | .480 |
| Shekel | .427 | .660 | .630 | .648 | .688 | .623 | .635 | .645 | .682 | .540 | .530 | .564 | .576 | .510 |
| Hart(6) | .265 | .348 | .338 | .340 | .354 | .347 | .334 | .330 | .333 | .297 | .304 | .266 | .301 | .262 |
| CPE | 190 | 55 | 100 | 100 | 66 | 100 | 100 | 100 | 91 | 118 | 133 | 137 | 120 | 138 |

We first note that the two offline algorithms (OfStages and OfIL) perform similarly across all three horizon settings. This suggests that there is limited benefit in these scenarios to using the more flexible IL schedules, which were primarily introduced for use in the online scheduling context. Comparing with the two online baselines (OnFCP and OnMEL), the offline algorithms perform significantly better. This may seem surprising at first because online policies should offer more flexibility than fixed offline schedules. However, the offline schedules purposefully wait for experiments to complete before starting up new experiments, which tends to improve the CPE values. To see this, the last row of Table 2 gives the average CPEs of each policy. Both OnFCP and OnMEL yield significantly lower CPEs compared to the offline algorithms, which correlates with their significantly larger regrets.

Finally, policy switching consistently outperforms other policies (excluding $h=\infty$) on the medium horizon setting and performs similarly in the other settings. This makes sense since the added flexibility of PS is not as critical for long and short horizons. For short horizons, there is less opportunity for scheduling choices and for longer horizons the scheduling problem is easier and hence the offline approaches are more competitive. In addition, looking at Table 2, we see that PS achieves a significantly higher CPE than offline approaches in the medium horizon, and is similar to them in the other horizons, again correlating with the regret. Further examination of the schedules produced by PS indicates that although it begins with the same number of labs as OfIL, PS often selects fewer labs in later steps if early experiments are completed sooner than expected, which leads to higher CPE and consequently better performance. Note that the variances of the proposed policies are very small which are shown in the supplementary materials.

## 7 Summary and Future Work

Motivated by real-world applications we introduced a novel setting for Bayesian optimization that incorporates a budget on the total time and number of experiments and allows for concurrent, stochastic-duration experiments. We considered offline and online approaches for scheduling experiments in this setting, relying on a black box function to intelligently select specific experiments at their scheduled start times. These approaches aimed to optimize a novel objective function, Cumulative Prior Experiments (CPE), which we empirically demonstrate to strongly correlate with performance on the original optimization problem. Our offline scheduling approaches significantly outperformed some natural baselines and our online approach of policy switching was the best overall performer.

For further work we plan to consider alternatives to CPE, which, for example, incorporate factors such as diminishing returns. We also plan to study further extensions to the experimental model for BO and also for active learning. For example, taking into account varying costs and duration distributions across labs and experiments. In general, we believe that there is much opportunity for more tightly integrating scheduling and planning algorithms into BO and active learning to more accurately model real-world conditions.

## Acknowledgments

The authors acknowledge the support of the NSF under grants IIS-0905678.

## Footnotes

[1] For simplicity our previous discussion of the IL scheduler did not consider states with ongoing experiments, which will occur here. To handle this the scheduler first considers using already executing labs taking into account how long they have been running. If more labs are required to ensure $p$-safeness new ones are added.

# References

[1] B. S. Anderson, A. Moore, and D. Cohn. A nonparametric approach to noisy and costly optimization. In *ICML*, 2000.

[2] J. Azimi, A. Fern, and X. Fern. Batch bayesian optimization via simulation matching. In *NIPS*, 2010.

[3] D. Bond and D. Lovley. Electricity production by geobacter sulfurreducens attached to electrodes. *Applications of Environmental Microbiology*, 69:1548–1555, 2003.

[4] E. Brochu, M. Cora, and N. de Freitas. A tutorial on Bayesian optimization of expensive cost functions, with application to active user modeling and hierarchical reinforcement learning. Technical Report TR-2009-23, Department of Computer Science, University of British Columbia, 2009.

[5] E. H. Burrows, W.-K. Wong, X. Fern, F. W. Chaplen, and R. L. Ely. Optimization of ph and nitrogen for enhanced hydrogen production by synechocystis sp. pcc 6803 via statistical and machine learning methods. *Biotechnology Progress*, 25:1009–1017, 2009.

[6] H. Chang, R. Givan, and E. Chong. Parallel rollout for online solution of partially observable markov decision processes. *Discrete Event Dynamic Systems*, 14:309–341, 2004.

[7] L. Dixon and G. Szeg. *The Global Optimization Problem: An Introduction Toward Global Optimization*. North-Holland, Amsterdam, 1978.

[8] D. Jones. A taxonomy of global optimization methods based on response surfaces. *Journal of Global Optimization*, pages 345–383, 2001.

[9] Z. Michalewicz. *Genetic algorithms + data structures = evolution programs (2nd, extended ed.)*. Springer-Verlag New York, Inc., New York, NY, USA, 1994.

[10] D. Park and J. Zeikus. Improved fuel cell and electrode designs for producing electricity from microbial degradation. *Biotechnol.Bioeng.*, 81(3):348–355, 2003.

